# Fast Transformation-Invariant Factor Analysis

**Anitha Kannan**[1]       **Nebojsa Jojic**[2]       **Brendan Frey**[1]

[1]University of Toronto, Toronto, Canada
{anitha, frey}@psi.utoronto.ca

[2]Microsoft Research, Redmond, WA, USA
jojic@microsoft.com

## Abstract

Dimensionality reduction techniques such as principal component analysis and factor analysis are used to discover a linear mapping between high dimensional data samples and points in a lower dimensional subspace. In [6], Jojic and Frey introduced mixture of transformation-invariant component analyzers (MTCA) that can account for global transformations such as translations and rotations, perform clustering and learn local appearance deformations by dimensionality reduction. However, due to enormous computational requirements of the EM algorithm for learning the model, $O(N^2)$ where $N$ is the dimensionality of a data sample, MTCA was not practical for most applications. In this paper, we demonstrate how fast Fourier transforms can reduce the computation to the order of $N\log N$. With this speedup, we show the effectiveness of MTCA in various applications - tracking, video textures, clustering video sequences, object recognition, and object detection in images.

## 1   Introduction

Dimensionality reduction techniques such as principal component analysis [7] and factor analysis [1] linearly map high dimensional data samples onto points in a lower dimensional subspace. In factor analysis, this mapping is defined by subspace origin $\mu$ and the subspace bases stored in the columns of the factor loading matrix, $\Lambda$. A mixture of factor analyzers learn to place the data into several learned subspaces. In computer vision, this approach has been widely used in face modeling for learning facial expressions (e.g. [2] , [12] ).

When the variability in the data is due, in part, to small transformations such as translations, scales and rotations, factor analyzer learns a linearized transformation manifold which is often sufficient ( [4], [11]). However, for large transformations present in the data, linear approximation is insufficient. For instance, a factor analyzer trained on a video sequence of a person walking tries to capture a linearized model of large translations (fig. 2a.) as opposed to learning local deformations such as motion of legs and hands (fig. 2c.).

In [6], it was shown that a discrete hidden transformation variable **T**, enables clustering and learning subspaces within clusters, invariant to global transformations. However, experiments were done on images of very low resolution due to enormous computational cost of EM algorithm used for learning the model. It is known that fast Fourier transform(FFT)

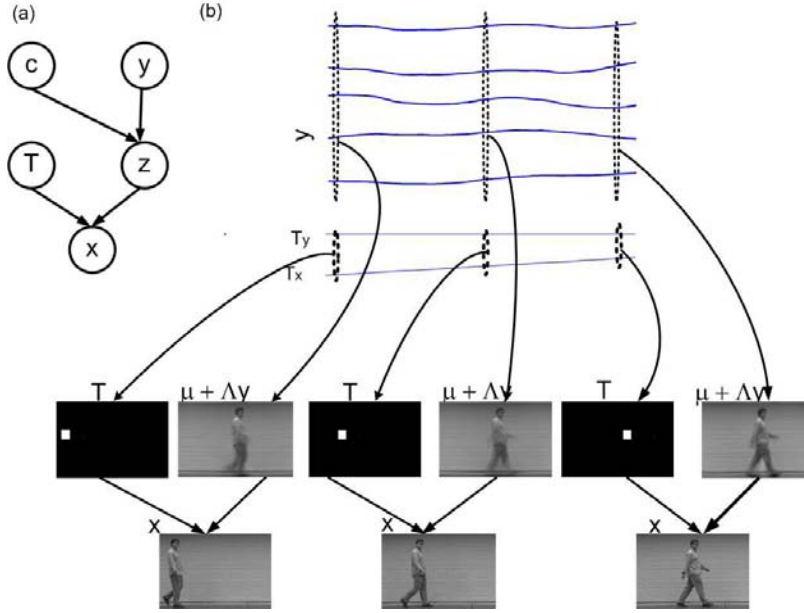

Figure 1: Mixture of transformed component analyzers (MTCA). (a) The generative model with cluster index c, subspace coordinates $\mathbf{y}$, latent image $\mathbf{z} = \boldsymbol{\mu} + \boldsymbol{\Lambda}_c\mathbf{y}$+noise; transformation $\mathbf{T}$ and generated final image $\mathbf{x} = \mathbf{Tz}$+noise; (b) An example of the generative process, where subspace coordinates $\mathbf{y}$, and image position $\mathbf{T}_x$, $\mathbf{T}_y$ are inferred from a captured video sequence $\mathbf{x}$

is very useful in dealing with transformations in images ( [3], [13]). The main purpose of this work is to show that under very mild assumptions, we can have an effective implementation of MTCA that reduces the complexity from $K^2|\mathcal{T}|\mathcal{N}$ to $K^2|\mathcal{T}|\log N$, where $K$ is the number of factors, N is the size of input, $\mathcal{T}$ is the set of all possible transformations. This means that for 256x256 images, the current implementation will be 4000 times faster.

We present experimental results showing the effectiveness of MTCA in various applications - tracking, video textures, clustering video sequences, object recognition and detection.

## 2  Fast inference and learning in MTCA

Mixture of transformation-invariant component analyzers (MTCA) is used for transformation-invariant clustering and learning a subspace representation within each cluster. The set of transformations, $\mathcal{T}$, to which the model is invariant is specified a priori. Fig. 1a. shows a generative model for MTCA. The vector $\mathbf{y}$ is a $K$ dimensional Gaussian $\mathcal{N}(\mathbf{0}, \mathbf{I})$ random variable. Cluster index, c is a C-valued discrete random variable with probability distribution, $\pi_c$. The $N$ dimensional ($N \gg K$)latent image, $\mathbf{z}$ has mean, $\boldsymbol{\mu}_c + \boldsymbol{\Lambda}_c\mathbf{y}$, and diagonal covariance, $\boldsymbol{\Phi}_c$; the $N$x$K$ matrix $\boldsymbol{\Lambda}_c$ is the factor loading matrix for class c. An observation $\mathbf{x}$ is obtained by applying a transformation $\mathbf{T} \in \mathcal{T}$, (with distribution $\rho_{\mathbf{T}}$) on the latent image $\mathbf{z}$ and adding independent Gaussian noise, $\boldsymbol{\Psi}$. Fig. 1b illustrates this generative process for a one class MTCA. The subspace coordinates $\mathbf{y}$, are used to generate a latent image, $\mathbf{z}$ (without noise), and the horizontal and vertical image position $\mathbf{T}_x$ and $\mathbf{T}_y$ are used to shift the latent image to obtain $\mathbf{x}$. In fact, $\mathbf{y}$, $\mathbf{T}_x$ and $\mathbf{T}_y$ shown in the figure are actually inferred from the captured video sequence $\mathbf{x}$ (see sec. 3). The joint distribution over all variables is [6],

$$p(\mathbf{y}, c, \mathbf{z}, \mathbf{T}, \mathbf{x}) = p(\mathbf{y})p(\mathbf{z}|\mathbf{y}, c)p(\mathbf{x}|\mathbf{z}, \mathbf{T})P(\mathbf{T})P(c)$$
$$= \mathcal{N}(\mathbf{y}; \mathbf{0}, \mathbf{I})\mathcal{N}(\mathbf{z}; \boldsymbol{\mu}_c + \boldsymbol{\Lambda}_c\mathbf{y}, \boldsymbol{\Phi}_c)\mathcal{N}(\mathbf{x}; \mathbf{Tz}, \boldsymbol{\Psi})\rho_{\mathbf{T}}\pi_c$$

(a)

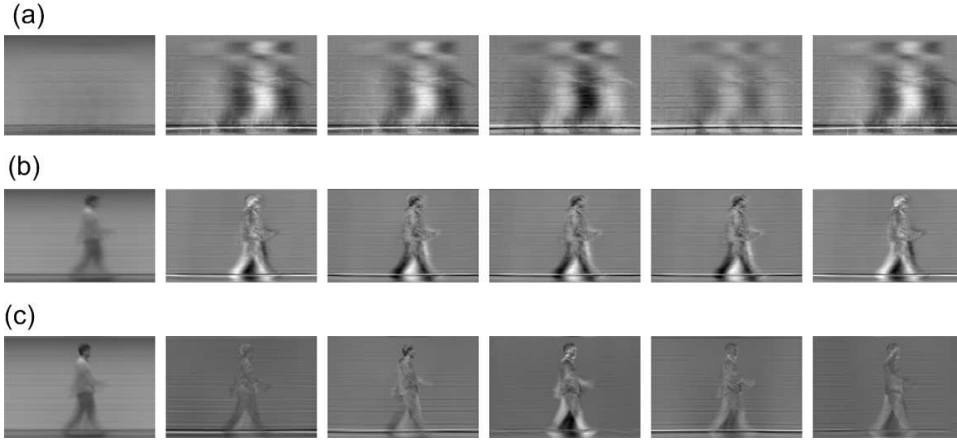

(b)

(c)

Figure 2: Means $\boldsymbol{\mu}$ and components in $\boldsymbol{\Lambda}$ learned using (a) FA, (b) FA applied on data normalized using a correlation tracker, and (c) transformed component analysis (TCA) applied directly on data.

Performing inference on transformations and class, $P(c, \mathbf{T}|\mathbf{x})$ requires evaluating the joint,

$$P(\mathbf{x}, c, \mathbf{T}) = P(\mathbf{x}|c, \mathbf{T})P(c)P(\mathbf{T}) = \mathcal{N}(\mathbf{x}; \mathbf{T}\boldsymbol{\mu}_c, \mathbf{T}(\boldsymbol{\Lambda}_c\boldsymbol{\Lambda}_c^T + \boldsymbol{\Phi}_c)^T + \boldsymbol{\Psi})\rho_{\mathbf{T}}\pi_c \quad (1)$$

The likelihood of $\mathbf{x}$ is

$$p(\mathbf{x}) = \sum_{c=1}^{C} \sum_{\mathbf{T} \in \mathcal{T}} \mathcal{N}(\mathbf{x}; \mathbf{T}\boldsymbol{\mu}_c, \mathbf{T}(\boldsymbol{\Lambda}_c\boldsymbol{\Lambda}_c^T + \boldsymbol{\Phi}_c)^T + \boldsymbol{\Psi})\rho_{\mathbf{T}}\pi_c \quad (2)$$

The parameters of the model are learned from i.i.d training examples by maximizing their likelihood ($\prod_j p(\mathbf{x}^{(j)})$) using an exact EM algorithm. The only inputs to the EM are the training examples, the number of factors, $K$, the number of clusters, $C$, and the set of all possible transformations, $\mathcal{T}$. Starting at a random initialization, EM algorithm for MTCA iterates between E step, where it probabilistically fills in for hidden variables by finding the exact posterior $p(\mathbf{y}, \mathbf{z}, c, \mathbf{T}|\mathbf{x})$ and M step in which it updates the parameters.

The likelihood of the data (eqn. 2) requires summing over all possible transformations and is very expensive. In fact, each of inference and update equations in [6] has a complexity of $K^2|\mathcal{T}|N$. In this section, we show how these equations can be derived and evaluated in Fourier domain at a considerably lower computational cost of $K^2|\mathcal{T}|\log N$. We focus on inferring the means of $p(\mathbf{y}|\mathbf{T}, c, \mathbf{x})$ and $p(\mathbf{z}|, c, \mathbf{x})$ as examples for efficient computation. Similar mathematical manipulations will result in the inferences provided in the appendix.

We assume that data is represented in a coordinate system in which transformations are discrete shifts with wrap-around. For translations, it is 2D rectangular grid, while for rotations and scales it is shifts in a radial grid (c.f. [3] [13]). We also assume that the post-transformation noise is isotropic, $\boldsymbol{\Psi} = \psi\mathbf{I}$, so that covariances matrices, $\text{COV}[\mathbf{z}|\mathbf{T}, c, \mathbf{x}]$ and $\text{COV}[\mathbf{y}|\mathbf{T}, c, \mathbf{x}]$ become independent of $\mathbf{T}$. In fact, for isotropic $\boldsymbol{\Psi}$, it is possible to preset $\psi$ (in our experiments we set it to .001). By presetting the sensor noise, $\psi$, to small value, if the actual value in the data is larger, it can be accounted for in $\boldsymbol{\Phi}$.

First, we describe the notation that simplifies expressions for transformation that corresponds to a shift in input coordinates. Define $\mathbf{i}$ to be an integer vector in the coordinate system in which input is measured. For 2D nxn image, $(N = n^2)$, $\mathbf{x}(\mathbf{i})$ is the $\mathbf{i}^{th}$ element where $\mathbf{i} \in \{(i_1, i_2) : i_1 = 1 \cdots n, i_2 = 1 \cdots n\}$. Vectors in the input coordinate system such as $\mathbf{x}, \boldsymbol{\mu}, \mathbf{z}$ are defined this way. For diagonal matrices such as $\boldsymbol{\Phi}$, $\boldsymbol{\Phi}(\mathbf{i})$ defines the element corresponding to pixel at coordinate $\mathbf{i}$. This notation enables treating transformations corresponding to a shift be represented as a vector $\mathbf{T}$ in the same system, so that a shift of $\mathbf{z}$ by $\mathbf{T}$ is represented as $\mathbf{z}(\mathbf{i} + \mathbf{T})$ such that $\mathbf{i} + \mathbf{T} = (i_1 + T_1 \bmod n, i_2 + T_2 \bmod n)$.

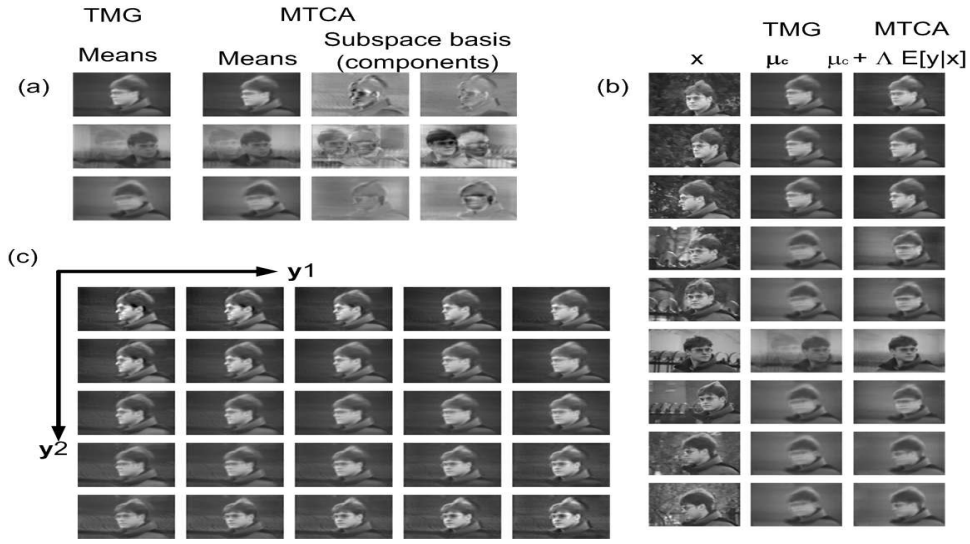

Figure 3: Transformation invariant clustering with and without a subspace model: (a) Parameters of a three-cluster TMG [6], and a three-cluster MTCA (b) Frames from the video sequence $\mathbf{x}$, corresponding TMG mean $\boldsymbol{\mu}_c$ and the object appearance $\boldsymbol{\mu}_c + \boldsymbol{\Lambda}_c \mathbf{y}$ in the corresponding subspace of MTCA; (c) An illustration of the role of components for the first class. Factor $\mathbf{y}_1$ tends to model lighting variation and $\mathbf{y}_2$ tends to model small out-of-plane rotations

In the appendix, we show that *all* expensive operations in inference and learning involve computing correlation $(\mathbf{p} \odot \mathbf{q})$, or convolution$(\mathbf{p} \otimes \mathbf{q})$. These operations is only $O(|\mathcal{T}| \log N)$, i.e. $O(N \log N)$ for all shifts in frequency domain, while it is $O(|\mathcal{T}|N)$ in the pixel domain. For notational ease, we represent $k^{th}$ column and row of a matrix, $\mathbf{V}$, by $(\mathbf{V})_{(:,k)}$ and $(\mathbf{V})_{(k,:)}$ respectively. Also, diag($\mathbf{V}$) extracts the diagonal elements of matrix $\mathbf{V}$ and $\mathbf{p} \circ \mathbf{q}$ defines an element wise product between $\mathbf{p}$ and $\mathbf{q}$.

In principal component analysis (PCA), where there is no noise, the data is projected to subspace through the projection matrix. Similarly, in MTCA, we can derive that when $\mathbf{TT}^T = \mathbf{I}$ and $\boldsymbol{\Psi} = \psi\mathbf{I}$, the projection matrix is $\mathbf{Q}_c = \mathbf{B_c}\boldsymbol{\Lambda}_c\mathbf{A_c}$ and it accounts for variances. $\mathbf{A_c} = (\boldsymbol{\Phi}_c + \boldsymbol{\Psi})^{-1}$ is the inverse of noise variances in input space, and $\mathbf{B_c} = (\boldsymbol{\Lambda}_c^T\mathbf{A_c}\boldsymbol{\Lambda}_c + \mathbf{I})^{-1}$ is the inverse of the noise variances in the projected subspace. The mean of subspace for a given $\mathbf{x}^{(j)}$, $c$, and $\mathbf{T}$ is obtained by subtracting the mean of the latent image $\boldsymbol{\mu}_c$ from the transformation-normalized $\mathbf{x}^{(j)}$ and applying the projection matrix: $E[\mathbf{y}|\mathbf{x}^{(j)}, \mathbf{T}, c] = \mathbf{Q_c}(\mathbf{T}^T\mathbf{x}^{(j)} - \boldsymbol{\mu}_c)$. For each factor, $\mathbf{y}_k$, it reduces to

$$E[\mathbf{y}_k|\mathbf{x}^{(j)}, \mathbf{T}, c] = \sum_{\mathbf{i}=1}^{N}(\mathbf{Q_c})_{k,\mathbf{i}}(\mathbf{x}^{(j)}(\mathbf{i} - \mathbf{T}) - \boldsymbol{\mu}_c(\mathbf{i})) = [(\mathbf{Q_c})_{k,:} \odot (\mathbf{x}^{(j)} - \boldsymbol{\mu}_c)]$$

As the summation over $\mathbf{i}$ for all $\mathbf{T}$ is a correlation, it can be efficiently computed for all $\mathbf{T}$ at the same time in the frequency domain in $N \log N$ time.

The inference on the latent image $\mathbf{z}$ is given by its expected value:

$$E[\mathbf{z}|c, \mathbf{x}^{(j)}] = \boldsymbol{\Omega}_c(\boldsymbol{\Lambda}_c\boldsymbol{\Lambda}_c^T + \boldsymbol{\Phi}_c)^{-1}\boldsymbol{\mu}_c + \boldsymbol{\Omega}_c\boldsymbol{\Psi}^{-1}\sum_{\mathbf{T}\in\mathcal{T}}P(\mathbf{T}|c, \mathbf{x}^{(j)})\mathbf{T}^T\mathbf{x}^{(j)}$$

where $\boldsymbol{\Omega}_c = \mathrm{COV}[\mathbf{z}|\mathbf{T}, c, \mathbf{x}^{(j)}]$ is independent of $\mathbf{x}^{(j)}$ and $\mathbf{T}$. The first term, dictated by the model can be easily computed. $\sum_{\mathbf{T}\in\mathcal{T}}P(\mathbf{T}|c, \mathbf{x}^{(j)})\mathbf{T}^T\mathbf{x}^{(j)}$ is a convolution of $\mathbf{x}^{(j)}$ with the probability map $P(\mathbf{T}|c, \mathbf{x}^{(j)})$ defined for all $\mathbf{T}$, as a particular element in the sum

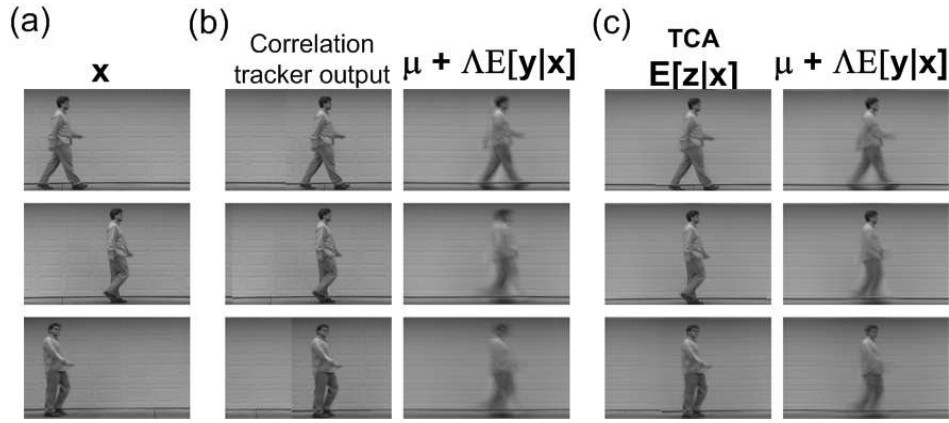

Figure 4: Comparison of FA applied on data normalized for translations using correlation tracker and TCA. (a)Frames from sequence. (b) shift normalized frames, using correlation-based tracker and $E[\mu + \Lambda y]$ obtained through factor analysis model. (c)$E[z|x]$ and $E[\mu + \Lambda y]$ for the TCA model.

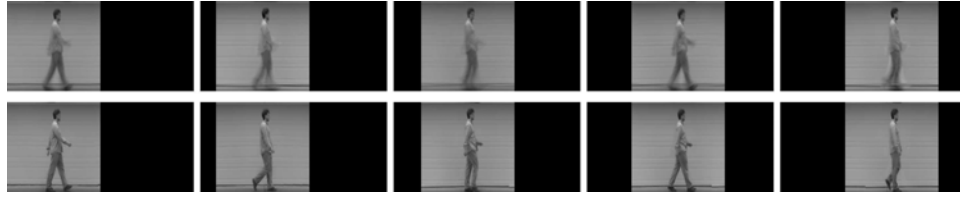

Figure 5: Simulated walk sequence synthesized after training an AR model on the subspace and image motion parameters. The sequence enlarged for better viewing of translations. The first row contains a few frames from the sequence simulated directly from the model. The second row contains a few frames from the video texture generated by picking the frames in the original sequence for which the recent subspace trajectory was similar to the one generated $y$ by the AR model.

is $\sum_{\mathbf{T}\in\mathcal{T}} P(\mathbf{T}|c,\mathbf{x}^{(j)})\mathbf{x}^{(j)}(\mathbf{i}-\mathbf{T})$. We can efficiently compute this sum for all $\mathbf{i}$:

$$E[\mathbf{z}|c,\mathbf{x}^{(j)}] = \mathbf{\Omega}_c(\mathbf{\Lambda}_c\mathbf{\Lambda}_c^T + \mathbf{\Phi}_c)^{-1}\boldsymbol{\mu}_c + \mathbf{\Omega}_c\mathbf{\Psi}^{-1}\left[P(\mathbf{T}|c,\mathbf{x}^{(j)})\otimes\mathbf{x}^{(j)}\right]$$

Note that multiplication with $N$x$N$ matrices above can be done efficiently by factorizing them and applying a sequence of vector multiplication from right to left.

## 3   Experimental Results

**Clustering face poses.** In Fig. 3b the first column shows examples from a video sequence of one person with different facial poses walking across cluttered background. We trained a transformation-invariant mixture of Gaussians (TMG) [6] with 3 clusters that learned means shown in Fig. 3b. TMG captures the three generic poses in the data.However, due to presence of lighting variations and small out-of-plane rotations in addition to big pose changes, it is difficult for TMG to learn these variations without many more classes.

We trained a MTCA model with 3 classes and 2 factors, initializing the parameters to those learned by TMG. Fig. 3a compares TMG means and components to those learned using MTCA. The MTCA model learns to capture small variations around the cluster means. For example, for the first cluster, the two subspace coordinates tend to model out-of-plane rotations and illumination changes (Fig. 3c). In Fig. 3b, we compare $E[\boldsymbol{\mu}_c + \mathbf{\Lambda}_c\mathbf{y}|\mathbf{x}]$, ($c = argmax_c P(c|\mathbf{x})$), of TMG and MTCA for various training examples, illustrating better tracking and appearance modelling of MTCA.

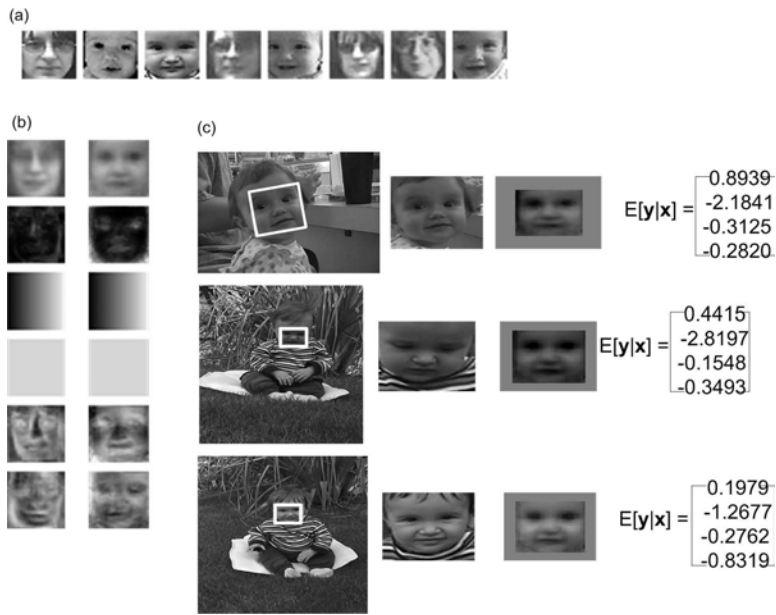

Figure 6: Clustering faces extracted from a personal database prepared using face detector. (a) Training examples (b) Means, variances and components for two classes learned using MTCA. (c) $1^{st}$ column contains several photos in which the detector [8] failed to find the face. $2^{nd}$ column contains central 100x100 portion of $E[\mathbf{z}|\mathbf{x}]$. $3^{rd}$ column contains central 100x100 portion of $\boldsymbol{\mu}_c + \boldsymbol{\Lambda}_c E[\mathbf{y}|\mathbf{x}]$.

**Modeling a walking person.** Fig. 4a. shows three 165x285 frames from a video sequence of a person walking. For effective summarization, we need to learn a compact representation for the dynamically and periodically changing hand and leg movements.

A regular PCA or FA will learn a representation that focuses more on learning linearized shifts, and less on the more interesting motion of hands and legs (Fig. 2a.). The traditional approach is to track the object using, for example,a correlation tracker and then learn the subspace model on normalized images. The parameters learned in this fashion are shown in Fig. 2b. Without previously learning a good model, the tracker fails to provide the perfect tracking necessary for precise subspace modelling of limb motion and thus the inferred subspace projection is blurred. (Fig. 2b).

As TCA performs tracking and learns appearance model at the same time, not only does it avoids the tracker initialization that plagues the "tracking first" approaches, but also provides perfectly aligned $E[\mathbf{z}|\mathbf{x}]$ and infers a much cleaner projection $E[\boldsymbol{\mu} + \boldsymbol{\Lambda}\mathbf{y}|\mathbf{x}]$.

The TCA model can be used to create video textures based on frame reshuffling similar to [10]. However, instead of shuffling frames based directly on pixel similarity, we use the subspace position $\mathbf{y}(t)$ and image position $\mathbf{T}^{(t)}$ generated from an AR process [9], and for each t find the best frame u in the original video $\mathbf{x}^{(u)}$ for which the window $E[\mathbf{y}|\mathbf{x}^{(u)}], E[\mathbf{y}|\mathbf{x}^{(u-1)}], ...E[\mathbf{y}|\mathbf{x}^{(u-9)}]$is the most similar to $\mathbf{y}^{(t)}, ..., \mathbf{y}^{(t-9)}$. Then, generated transformation $\mathbf{T}$ is applied on the normalized image $E[\mathbf{z}|\mathbf{x}^{(u)}]$. The result is shown in fig. 5b and contains a bit sharper images than the ones simulated directly from the generative model, fig. 5a. We let the simulated walk last longer than in the original sequence letting MTCA live on twice as wide frames.

**Clustering and face recognition** We used a standard face detector [8] to obtain 85 32x32 images of faces of 2 persons, from a personal photo database of a mother and her daughter. In fig. 6a. we present examples from the training set.

We learned a MTCA model with 2 classes and 4 factors. To model global lighting variation, we preset one of the factors to be uniform at .01 (see fig. 6b.). This handles linearized version of ambient lighting condition. We also preset another factor to be smoothly varying in brightness (see fig. 6b.) to capture side illumination. The other two components are learned and they model slight appearance deformation such as facial expressions. The model learned to cluster faces in the training example with 94.12% accuracy.

An interesting application is to use the learned representation of the faces to detect and recognize faces in the original photos. For instance, the face detector did not recognize faces in many photographs (for eg.,fig. 6c), which we were able to using the learned model (fig. 6c). We increased the resolution of model parameters $\boldsymbol{\mu}_c, \boldsymbol{\Lambda}_c, \boldsymbol{\Phi}_c$ to match the resolution of photos (640x480), padding around the original parameters with uniform mean, zero factors and high variance. Then, we performed inference, inferring most likely class,c, most likely,$\mathbf{T}$ for that class and $E[\mathbf{z}|\mathbf{x}, c]$. We also incorporated 3 rotations and 4 scales as possible transformations, in addition to all possible shifts. In fig. 6c , we present three examples which were not in the training set and the face detector we used failed. In all three cases MTCA detected *and recognized* the face correctly as belonging to class 2.

## 4 Conclusion

Mixture of transformation-invariant component analyzers is a technique for modeling visual data that finds major clusters and major transformations in the data and learns subspace models of appearance. In this paper, we have described how a fast implementation of learning the model is possible through efficient use of identities from matrix algebra and the use of fast Fourier transforms.

**Appendix: EM updates** Before performing inference in Estep, we pre-compute,for all classes the quantities that are independent of training examples:

$\mathbf{A_c} = (\boldsymbol{\Phi}_c + \boldsymbol{\Psi})^{-1} \qquad \mathbf{B_c} = (\boldsymbol{\Lambda}_c^T \mathbf{A_c} \boldsymbol{\Lambda}_c + \mathbf{I})^{-1} \qquad \mathbf{D_c} = (\boldsymbol{\Phi}_c^{-1} + \boldsymbol{\Psi}^{-1})^{-1} \qquad \mathbf{Q_c} = \mathbf{B_c} \boldsymbol{\Lambda}_c^T \mathbf{A_c}$

$\mathbf{F_c} = \mathbf{D_c} \boldsymbol{\Phi}_c^{-1} \boldsymbol{\Lambda}_c \mathbf{B_c} \qquad \mathbf{N_c} = (\boldsymbol{\Lambda}_c^T \boldsymbol{\Phi}_c^{-1} \boldsymbol{\Lambda}_c + \mathbf{I})^{-1} \qquad \mathbf{M_c} = \boldsymbol{\mu}_c^T \boldsymbol{\Phi}_c^{-1} \boldsymbol{\Lambda}_c \qquad \mathbf{S_c} = \boldsymbol{\Lambda}_c^T \boldsymbol{\Phi}_c^{-1} \mathbf{D_c}$

$\mathbf{g_c} = \boldsymbol{\Psi}(\mathbf{A_c} - \mathbf{A_c} \boldsymbol{\Lambda}_c \mathbf{B_c} \boldsymbol{\Lambda}_c^T \mathbf{A_c}) \boldsymbol{\mu}_c \qquad \mathbf{H_c} = (\mathbf{D_c} \boldsymbol{\Phi}_c^{-1} \boldsymbol{\Lambda}_c + \mathbf{F_c} \mathbf{S_c} \boldsymbol{\Phi}_c^{-1} \boldsymbol{\Lambda}_c)^T$

For each training example $\mathbf{x}^{(j)}$, $E[\mathbf{y}_k|\mathbf{x}^{(j)}, \mathbf{T}, c] = \left[ (\mathbf{Q_c})_{k,:} \odot (\mathbf{x}^{(j)} - \boldsymbol{\mu}_c) \right]$ is evaluated and saved.

Computing posteriors over $\mathbf{T}$ and c, requires evaluating $p(\mathbf{x}^{(j)}|\mathbf{T}, c)$ (eqn. 1). To compute this distribution, we require the determinant of covariance matrix, $\mathsf{COV}[\mathbf{x}^{(j)}|\mathbf{T}, c]$ and the Mahalanobis distance between input and the transformed latent image. The determinant is simplified to

$|\mathsf{COV}[\mathbf{x}^{(j)}|\mathbf{T}, c]| = |\boldsymbol{\Phi}_c + \boldsymbol{\Psi}||\boldsymbol{\Lambda}_c^T (\boldsymbol{\Phi}_c + \boldsymbol{\Psi})^{-1} \boldsymbol{\Lambda}_c + \mathbf{I}|$

The Mahalanobis distance between $\mathbf{x}^{(j)}$ and the latent image is

$(\mathbf{T}^T \mathbf{x}^{(j)})^T \mathbf{A_c} \mathbf{T}^T \mathbf{x}^{(j)} - 2\boldsymbol{\mu}_c^T \mathbf{A_c} \mathbf{T}^T \mathbf{x}^{(j)} + \boldsymbol{\mu}_c^T \mathbf{A_c} \boldsymbol{\mu}_c - E[\mathbf{y}|\mathbf{T}, \mathbf{x}^{(j)}]^T \mathbf{B_c}^{-1} E[\mathbf{y}|\mathbf{T}, \mathbf{x}^{(j)}]$

$$a(k, \mathbf{T}) = (\mathbf{W_c})_{k,:} \odot \mathbf{x}^{(j)} \qquad (\mathbf{L_{T,c}})_{:,k} = \mathbf{g_c}^T \boldsymbol{\Phi}_c^{-1} \boldsymbol{\Lambda}_c + \boldsymbol{\Psi}^{-1}((\mathbf{H_c})_{k,:} \odot \mathbf{x}^{(j)})$$

$$E[diag(\mathbf{zz}^T|c, \mathbf{x}^{(j)})] = diag(\mathbf{D_c}) - diag(\mathbf{D_c} \mathbf{D_c} \boldsymbol{\Phi}_c^{-1} \boldsymbol{\Phi}_c^{-1}) \circ \sum_{k=1}^{K} (\boldsymbol{\Lambda}_c)_{:,k} \circ (\boldsymbol{\Lambda}_c \mathbf{B_c})_{:,k}$$

$$+ diag(\mathbf{D_c}^2 \boldsymbol{\Psi}^{-2}) \circ (P(\mathbf{T}|c, \mathbf{x}^{(j)}) \otimes (\mathbf{x}^{(j)})^2)$$

$$+ diag(\mathbf{D_c}^2 \boldsymbol{\Phi}_c^{-2} \boldsymbol{\Psi}^{-2}) \circ \Bigg\{ -2 * \sum_{k=1}^{K} (\mathbf{P_c})_{:,k} \Big( (P(\mathbf{T}|\mathbf{x}^{(j)}) a(k, \mathbf{T})) \otimes \mathbf{x}^{(j)} \Big)$$

$$\sum_{k=1}^{K} (\mathbf{P_c})_{:,k} \Big[ \sum_{l=1}^{K} (\mathbf{P_c})_{:,l} \sum_{\mathbf{T} \in \mathcal{T}} P(\mathbf{T}|\mathbf{x}^{(j)}) a(k, \mathbf{T}) a(l, \mathbf{T}) \Big] \Bigg\}$$

The summation over $\mathbf{T}$ takes only $|\mathcal{T}|$ time, after $a(k, \mathbf{T}) \forall k$ and $\mathbf{T}$ is computed in $K|\mathcal{T}|\log N$ time.

$$E[\mathbf{z}|\mathbf{x}^{(j)}, c] = \mathbf{g_c} + \mathbf{D_c} \boldsymbol{\Psi}^{-1} \Big[ P(\mathbf{T}|c, \mathbf{x}^{(j)}) \otimes \mathbf{x}^{(j)} \Big] + \mathbf{F_c} \Big( \mathbf{S_c} \boldsymbol{\Psi}^{-1} \Big[ P(\mathbf{T}|c, \mathbf{x}^{(j)}) \otimes \mathbf{x}^{(j)} \Big] \Big)$$

$$E[(\mathbf{y}\mathbf{y}^T)_{i,j}|\mathbf{x}^{(j)},c] \;=\; (\mathbf{B_c})_{i,j} + \sum_{\mathbf{T}\in\mathcal{T}} \left(P(\mathbf{T}|c,\mathbf{x}^{(j)}) \circ E[\mathbf{y}_i|\mathbf{x}^{(j)},\mathbf{T},c] \circ E[\mathbf{y}_j|\mathbf{x}^{(j)},\mathbf{T},c]\right)$$

$$E[diag(\mathbf{z}-\boldsymbol{\mu}_c-\boldsymbol{\Lambda}_c\mathbf{y})(\mathbf{z}-\boldsymbol{\mu}_c-\boldsymbol{\Lambda}_c\mathbf{y})^T|\mathbf{x}^{(j)},c] = E[diag(\mathbf{z}\mathbf{z}^T|c,\mathbf{x}^{(j)})] + \boldsymbol{\mu}_c\circ\boldsymbol{\mu}_c$$

$$+ \sum_{k=1}^{K} (\boldsymbol{\Lambda}_c)_{:,k} \circ \left((\boldsymbol{\Lambda}_c E[\mathbf{y}\mathbf{y}^T|\mathbf{x}^{(j)},c])_{:,k} - 2E[\mathbf{z}\mathbf{y}^T|c,\mathbf{x}^{(j)}]\right)$$

$$- 2\boldsymbol{\mu}_c \circ E[\mathbf{z}|c,\mathbf{x}^{(j)}] + 2(\boldsymbol{\Lambda}_c E[\mathbf{y}|c,\mathbf{x}^{(j)}])\circ\boldsymbol{\mu}_c$$

$$E[\mathbf{z}\mathbf{z}^T|c,\mathbf{x}^{(j)}]\boldsymbol{\Phi}_c^{-1}\boldsymbol{\Lambda}_c = \mathbf{H_c} + \mathbf{g_c}\sum_{k=1}^{K} P(\mathbf{T}|c,\mathbf{x}^{(j)})(\mathbf{L_{T,c}})_{:,k}$$

$$+ (\mathbf{D_c} + \mathbf{F_c}\mathbf{S_c})\boldsymbol{\Psi}^{-1}\left(\sum_{k=1}^{K} P(\mathbf{T}|c,\mathbf{x}^{(j)})(\mathbf{L_{T,c}})_{:,k}\otimes\mathbf{x}^{(j)}\right)$$

$$E[\mathbf{z}\mathbf{y}^T|c,\mathbf{x}^{(j)}] = \left[E[\mathbf{z}\mathbf{z}^T|c,\mathbf{x}^{(j)}]\boldsymbol{\Phi}_c^{-1}\boldsymbol{\Lambda}_c - E[\mathbf{z}|c,\mathbf{x}^{(j)}]\boldsymbol{\mu}_c^T\boldsymbol{\Phi}_c^{-1}\boldsymbol{\Lambda}_c\right]\mathbf{N_c}$$

Defining, $\left\langle \aleph \right\rangle = \frac{\sum_{j=1}^{J} P(c|\mathbf{x}^{(j)})\aleph}{\sum_{j=1}^{J} P(c|\mathbf{x}^{(j)})}$, in the Mstep, parameters are updated according to

$$\boldsymbol{\mu}_c \leftarrow \left\langle E[\mathbf{z}|c,\mathbf{x}^{(j)}] - \boldsymbol{\Lambda}_c E[\mathbf{y}|c,\mathbf{x}^{(j)}]\right\rangle$$

$$\boldsymbol{\Phi}_c \leftarrow \left\langle E[diag(\mathbf{z}-\boldsymbol{\mu}_c-\boldsymbol{\Lambda}_c\mathbf{y})(\mathbf{z}-\boldsymbol{\mu}_c-\boldsymbol{\Lambda}_c\mathbf{y})^T|\mathbf{x}^{(j)},c]\right\rangle$$

$$\boldsymbol{\Lambda}_c \leftarrow \left\langle E[\mathbf{z}\mathbf{y}^T|c,\mathbf{x}^{(j)}] - \boldsymbol{\mu}_c E[\mathbf{y}^T|c,\mathbf{x}^{(j)}]\right\rangle\left\langle E[\mathbf{y}\mathbf{y}^T|c,\mathbf{x}^{(j)}]\right\rangle^{-1}$$

# References

[1] Everitt, B.S. *An Introduction to Latent variable models* Chapman and Hall, New York NY 1984

[2] Frey, B.J. , Colmenarez, A. & Huang, T.S. Mixtures of local linear subspaces for face recognition. In *Proceedings of the IEEE Conference on Computer Vision and Pattern Recognition*. 1998 IEEE Computer Society Press: Los Alamitos, CA.

[3] Frey, B.J. & Jojic, N. Fast, large-scale transformation-invariant clustering. In *Advances in Neural Information Processing Systems 14*. Cambridge, MA: MIT Press 2002

[4] Ghahramani, Z. & Hinton, G. The EM Algorithm for Mixtures of Factor Analyzers *University of Toronto Technical Report* CRG-TR-96-1, 1996

[5] Hinton, G., Dayan, P. & Revow, M. Modeling the manifolds of images of handwritten digits In *IEEE Transactions on Neural Networks* 1997

[6] Jojic, N. & Frey, B.J. Topographic transformation as a discrete latent variable In *Advances in Neural Information Processing Systems 13*. Cambridge, MA: MIT Press 1999

[7] Jolliffe,I.T. *Principal Component Analysis* Springer-Verlag, New York NY, 1986.

[8] Li, S.Z., Zhu.L , Zhang, Z.Q. & Zhang,H.J. Learning to Detect Multi-View Faces in Real-Time In *Proceedings of the 2nd International Conference on Development and Learning*, June, 2002.

[9] Neumaier, A. & Schneider,T. Estimation of parameters and eigenmodes of multivariate autoregressive models In *ACM Transactions on Math Software* 2001.

[10] Schdl,A. Szeliski,R.,Salesin,D.& Irfan Essa Video textures In *Proceedings of SIGGRAPH*2000

[11] Simard, P. , LeCun, Y. & Denker, J. Efficient pattern recognition using a new transformation distance In *Advances in Neural Information Processing Systems* 1993

[12] Turk, M. & Pentland, A. Face recognition using eigenfaces In *Proceedings of IEEE Conference on Computer Vision and Pattern Recognition* Maui, Hawaii, 1991

[13] Wolberg,G. & Zokai,S. Robust image registration using log-polar transform In *Proceedings IEEE Intl. Conference on Image Processing, Canada* 2000.
